# Actor-Critic Algorithms

**Vijay R. Konda**    **John N. Tsitsiklis**
Laboratory for Information and Decision Systems,
Massachusetts Institute of Technology,
Cambridge, MA, 02139.
*konda@mit.edu, jnt@mit.edu*

## Abstract

We propose and analyze a class of actor-critic algorithms for simulation-based optimization of a Markov decision process over a parameterized family of randomized stationary policies. These are two-time-scale algorithms in which the critic uses TD learning with a linear approximation architecture and the actor is updated in an approximate gradient direction based on information provided by the critic. We show that the features for the critic should span a subspace prescribed by the choice of parameterization of the actor. We conclude by discussing convergence properties and some open problems.

## 1  Introduction

The vast majority of Reinforcement Learning (RL) [9] and Neuro-Dynamic Programming (NDP) [1] methods fall into one of the following two categories:

(a) Actor-only methods work with a parameterized family of policies. The gradient of the performance, with respect to the actor parameters, is directly estimated by simulation, and the parameters are updated in a direction of improvement [4, 5, 8, 13]. A possible drawback of such methods is that the gradient estimators may have a large variance. Furthermore, as the policy changes, a new gradient is estimated independently of past estimates. Hence, there is no "learning," in the sense of accumulation and consolidation of older information.

(b) Critic-only methods rely exclusively on value function approximation and aim at learning an approximate solution to the Bellman equation, which will then hopefully prescribe a near-optimal policy. Such methods are indirect in the sense that they do not try to optimize directly over a policy space. A method of this type may succeed in constructing a "good" approximation of the value function, yet lack reliable guarantees in terms of near-optimality of the resulting policy.

Actor-critic methods aim at combining the strong points of actor-only and critic-only methods. The critic uses an approximation architecture and simulation to learn a value function, which is then used to update the actor's policy parameters

in a direction of performance improvement. Such methods, as long as they are gradient-based, may have desirable convergence properties, in contrast to critic-only methods for which convergence is guaranteed in very limited settings. They hold the promise of delivering faster convergence (due to variance reduction), when compared to actor-only methods. On the other hand, theoretical understanding of actor-critic methods has been limited to the case of lookup table representations of policies [6].

In this paper, we propose some actor-critic algorithms and provide an overview of a convergence proof. The algorithms are based on an important observation. Since the number of parameters that the actor has to update is relatively small (compared to the number of states), the critic need not attempt to compute or approximate the exact value function, which is a high-dimensional object. In fact, we show that the critic should ideally compute a certain "projection" of the value function onto a low-dimensional subspace spanned by a set of "basis functions," that are *completely determined* by the parameterization of the actor. Finally, as the analysis in [11] suggests for TD algorithms, our algorithms can be extended to the case of arbitrary state and action spaces as long as certain ergodicity assumptions are satisfied.

We close this section by noting that ideas similar to ours have been presented in the simultaneous and independent work of Sutton et al. [10].

## 2 Markov decision processes and parameterized family of RSP's

Consider a Markov decision process with finite state space $S$, and finite action space $A$. Let $g : S \times A \to \mathbb{R}$ be a given cost function. A *randomized stationary policy* (RSP) is a mapping $\mu$ that assigns to each state $x$ a probability distribution over the action space $A$. We consider a set of randomized stationary policies $\mathbb{P} = \{\mu_\theta; \theta \in \mathbb{R}^n\}$, parameterized in terms of a vector $\theta$. For each pair $(x, u) \in S \times A$, $\mu_\theta(x, u)$ denotes the probability of taking action $u$ when the state $x$ is encountered, under the policy corresponding to $\theta$. Let $p_{xy}(u)$ denote the probability that the next state is $y$, given that the current state is $x$ and the current action is $u$. Note that under any RSP, the sequence of states $\{X_n\}$ and of state-action pairs $\{X_n, U_n\}$ of the Markov decision process form Markov chains with state spaces $S$ and $S \times A$, respectively. We make the following assumptions about the family of policies $\mathbb{P}$.

(A1) For all $x \in S$ and $u \in A$ the map $\theta \mapsto \mu_\theta(x, u)$ is twice differentiable with bounded first, second derivatives. Furthermore, there exists a $\mathbb{R}^n$-valued function $\psi_\theta(x, u)$ such that $\nabla \mu_\theta(x, u) = \mu_\theta(x, u)\psi_\theta(x, u)$ where the mapping $\theta \mapsto \psi_\theta(x, u)$ is bounded and has first bounded derivatives for any fixed $x$ and $u$.

(A2) For each $\theta \in \mathbb{R}^n$, the Markov chains $\{X_n\}$ and $\{X_n, U_n\}$ are irreducible and aperiodic, with stationary probabilities $\pi_\theta(x)$ and $\eta_\theta(x, u) = \pi_\theta(x)\mu_\theta(x, u)$, respectively, under the RSP $\mu_\theta$.

In reference to Assumption (A1), note that whenever $\mu_\theta(x, u)$ is nonzero we have

$$\psi_\theta(x, u) = \frac{\nabla \mu_\theta(x, u)}{\mu_\theta(x, u)} = \nabla \ln \mu_\theta(x, u).$$

Consider the average cost function $\lambda : \mathbb{R}^n \mapsto \mathbb{R}$, given by

$$\lambda(\theta) = \sum_{x \in S, u \in A} g(x, u)\eta_\theta(x, u).$$

We are interested in minimizing $\lambda(\theta)$ over all $\theta$. For each $\theta \in \mathbb{R}^n$, let $V_\theta : S \mapsto \mathbb{R}$ be the "differential" cost function, defined as solution of Poisson equation:

$$\lambda(\theta) + V_\theta(x) = \sum_{u \in A} \mu_\theta(x, u) \left[ g(x, u) + \sum_y p_{xy}(u) V_\theta(y) \right].$$

Intuitively, $V_\theta(x)$ can be viewed as the "disadvantage" of state $x$: it is the expected excess cost – on top of the average cost – incurred if we start at state $x$. It plays a role similar to that played by the more familiar value function that arises in total or discounted cost Markov decision problems. Finally, for every $\theta \in \mathbb{R}^n$, we define the $q$-function $q_\theta : S \times A \to \mathbb{R}$, by

$$q_\theta(x, u) = g(x, u) - \lambda(\theta) + \sum_y p_{xy}(u) V_\theta(y).$$

We recall the following result, as stated in [8]. (Different versions of this result have been established in [3, 4, 5].)

**Theorem 1.**

$$\frac{\partial}{\partial \theta_i} \lambda(\theta) = \sum_{x, u} \eta_\theta(x, u) q_\theta(x, u) \psi_\theta^i(x, u) \tag{1}$$

*where $\psi_\theta^i(x, u)$ stands for the ith component of $\psi_\theta$.*

In [8], the quantity $q_\theta(x, u)$ in the above formula is interpreted as the expected excess cost incurred over a certain renewal period of the Markov chain $\{X_n, U_n\}$, under the RSP $\mu_\theta$, and is then estimated by means of simulation, leading to actor-only algorithms. Here, we provide an alternative interpretation of the formula in Theorem 1, as an inner product, and thus derive a different set of algorithms, which readily generalize to the case of an infinite space as well.

For any $\theta \in \mathbb{R}^n$, we define the inner product $\langle \cdot, \cdot \rangle_\theta$ of two real valued functions $q_1, q_2$ on $S \times A$, viewed as vectors in $\mathbb{R}^{|S||A|}$, by

$$\langle q_1, q_2 \rangle_\theta = \sum_{x, u} \eta_\theta(x, u) q_1(x, u) q_2(x, u).$$

With this notation we can rewrite the formula (1) as

$$\frac{\partial}{\partial \theta_i} \lambda(\theta) = \langle q_\theta, \psi_\theta^i \rangle_\theta, \qquad i = 1, \ldots, n.$$

Let $\| \cdot \|_\theta$ denote the norm induced by this inner product on $\mathbb{R}^{|S||A|}$. For each $\theta \in \mathbb{R}^n$ let $\Psi_\theta$ denote the span of the vectors $\{\psi_\theta^i; \ 1 \leq i \leq n\}$ in $\mathbb{R}^{|S||A|}$. (This is same as the set of all functions $f$ on $S \times A$ of the form $f(x, u) = \sum_{i=1}^n \alpha_i \psi_\theta^i(x, u)$, for some scalars $\alpha_1, \ldots, \alpha_n$.)

Note that although the gradient of $\lambda$ depends on the $q$-function, which is a vector in a possibly very high dimensional space $\mathbb{R}^{|S||A|}$, the dependence is only through its inner products with vectors in $\Psi_\theta$. Thus, instead of "learning" the function $q_\theta$, it would suffice to learn the projection of $q_\theta$ on the subspace $\Psi_\theta$.

Indeed, let $\Pi_\theta : \mathbb{R}^{|S||A|} \mapsto \Psi_\theta$ be the projection operator defined by

$$\Pi_\theta q = \arg \min_{\hat{q} \in \Psi_\theta} \| q - \hat{q} \|_\theta.$$

Since

$$\langle q_\theta, \psi_\theta \rangle_\theta = \langle \Pi_\theta q_\theta, \psi_\theta \rangle_\theta, \tag{2}$$

it is enough to compute the projection of $q_\theta$ onto $\Psi_\theta$.

# 3   Actor-critic algorithms

We view actor critic-algorithms as stochastic gradient algorithms on the parameter space of the actor. When the actor parameter vector is $\theta$, the job of the critic is to compute an approximation of the projection $\Pi_\theta q_\theta$ of $q_\theta$ onto $\Psi_\theta$. The actor uses this approximation to update its policy in an approximate gradient direction. The analysis in [11, 12] shows that this is precisely what TD algorithms try to do, i.e., to compute the projection of an exact value function onto a subspace spanned by feature vectors. This allows us to implement the critic by using a TD algorithm. (Note, however, that other types of critics are possible, e.g., based on batch solution of least squares problems, as long as they aim at computing the same projection.)

We note some minor differences with the common usage of TD. In our context, we need the projection of $q$-functions, rather than value functions. But this is easily achieved by replacing the Markov chain $\{x_t\}$ in [11, 12] by the Markov chain $\{X_n, U_n\}$. A further difference is that [11, 12] assume that the control policy and the feature vectors are fixed. In our algorithms, the control policy as well as the features need to change as the actor updates its parameters. As shown in [6, 2], this need not pose any problems, as long as the actor parameters are updated on a slower time scale.

We are now ready to describe two actor-critic algorithms, which differ only as far as the critic updates are concerned. In both variants, the critic is a TD algorithm with a linearly parameterized approximation architecture for the $q$-function, of the form

$$Q_r^\theta(x, u) = \sum_{j=1}^m r^j \phi_\theta^j(x, u),$$

where $r = (r^1, \ldots, r^m) \in \mathbb{R}^m$ denotes the parameter vector of the critic. The features $\phi_\theta^j$, $j = 1, \ldots, m$, used by the critic are dependent on the actor parameter vector $\theta$ and are chosen such that their span in $\mathbb{R}^{|S||A|}$, denoted by $\Phi_\theta$, contains $\Psi_\theta$. Note that the formula (2) still holds if $\Pi_\theta$ is redefined as projection onto $\Phi_\theta$ as long as $\Phi_\theta$ contains $\Psi_\theta$. The most straightforward choice would be to let $m = n$ and $\phi_\theta^i = \psi_\theta^i$ for each $i$. Nevertheless, we allow the possibility that $m > n$ and $\Phi_\theta$ properly contains $\Psi_\theta$, so that the critic uses more features than that are actually necessary. This added flexibility may turn out to be useful in a number of ways:

1.  It is possible for certain values of $\theta$, the features $\psi_\theta$ are either close to zero or are almost linearly dependent. For these values of $\theta$, the operator $\Pi_\theta$ becomes ill-conditioned and the algorithms can become unstable. This might be avoided by using richer set of features $\psi_\theta^i$.

2.  For the second algorithm that we propose (TD($\alpha$) $\alpha < 1$) critic can only compute approximate - rather than exact - projection. The use of additional features can result in a reduction of the approximation error.

Along with the parameter vector $r$, the critic stores some auxiliary parameters: these are a (scalar) estimate $\lambda$, of the average cost, and an $m$-vector $z$ which represents Sutton's eligibility trace [1, 9]. The actor and critic updates take place in the course of a simulation of a single sample path of the controlled Markov chain. Let $r_k, z_k, \lambda_k$ be the parameters of the critic, and let $\theta_k$ be the parameter vector of the actor, at time $k$. Let $(X_k, U_k)$ be the state-action pair at that time. Let $X_{k+1}$ be the new state, obtained after action $U_k$ is applied. A new action $U_{k+1}$ is generated according to the RSP corresponding to the actor parameter vector $\theta_k$. The critic carries out an update similar to the average cost temporal-difference method of [12]:

$$\lambda_{k+1} = \lambda_k + \gamma_k(g(X_k, U_k) - \lambda_k),$$

$$r_{k+1} = r_k + \gamma_k \Big( g(X_k, U_k) - \lambda_k + Q_{r_k}^{\theta_k}(X_{k+1}, U_{k+1}) - Q_{r_k}^{\theta_k}(X_k, U_k) \Big) z_k.$$

(Here, $\gamma_k$ is a positive stepsize parameter.) The two variants of the critic use different ways of updating $z_k$:

*TD(1) Critic:* Let $x^*$ be a state in $S$.

$$
\begin{aligned}
z_{k+1} &= z_k + \phi_{\theta_k}(X_{k+1}, U_{k+1}), \qquad \text{if } X_{k+1} \neq x^*, \\
&= \phi_{\theta_k}(X_{k+1}, U_{k+1}), \qquad \text{otherwise.}
\end{aligned}
$$

*TD($\alpha$) Critic,* $0 \leq \alpha < 1$:

$$z_{k+1} = \alpha z_k + \phi_{\theta_k}(X_{k+1}, U_{k+1}).$$

*Actor:* Finally, the actor updates its parameter vector by letting

$$\theta_{k+1} = \theta_k - \beta_k \Gamma(r_k) Q_{r_k}^{\theta_k}(X_{k+1}, U_{k+1}) \psi_{\theta_k}(X_{k+1}, U_{k+1}).$$

Here, $\beta_k$ is a positive stepsize and $\Gamma(r_k) > 0$ is a normalization factor satisfying:

(A3) $\Gamma(\cdot)$ is Lipschitz continuous.

(A4) There exists $C > 0$ such that

$$\Gamma(r) \leq \frac{C}{1 + \|r\|}.$$

The above presented algorithms are only two out of many variations. For instance, one could also consider "episodic" problems in which one starts from a given initial state and runs the process until a random termination time (at which time the process is reinitialized at $x^*$), with the objective of minimizing the expected cost until termination. In this setting, the average cost estimate $\lambda_k$ is unnecessary and is removed from the critic update formula. If the critic parameter $r_k$ were to be reinitialized each time that $x^*$ is entered, one would obtain a method closely related to Williams' REINFORCE algorithm [13]. Such a method does not involve any value function learning, because the observations during one episode do not affect the critic parameter $r$ during another episode. In contrast, in our approach, the observations from all past episodes affect current critic parameter $r$, and in this sense critic is "learning". This can be advantageous because, as long as $\theta$ is slowly changing, the observations from recent episodes carry useful information on the $q$-function under the current policy.

## 4 Convergence of actor-critic algorithms

Since our actor-critic algorithms are gradient-based, one cannot expect to prove convergence to a globally optimal policy (within the given class of RSP's). The best that one could hope for is the convergence of $\nabla\lambda(\theta)$ to zero; in practical terms, this will usually translate to convergence to a local minimum of $\lambda(\theta)$. Actually, because the $TD(\alpha)$ critic will generally converge to an approximation of the desired projection of the value function, the corresponding convergence result is necessarily weaker, only guaranteeing that $\nabla\lambda(\theta_k)$ becomes small (infinitely often). Let us now introduce some further assumptions.

(A5) For each $\theta \in \mathbb{R}^n$, we define an $m \times m$ matrix $G(\theta)$ by

$$G(\theta) = \sum_{x,u} \eta_\theta(x,u)\phi_\theta(x,u)\phi_\theta(x,u)^T.$$

We assume that $G(\theta)$ is uniformly positive definite, that is, there exists some $\epsilon_1 > 0$ such that for all $r \in \mathbb{R}^m$ and $\theta \in \mathbb{R}^n$

$$r^T G(\theta) r \geq \epsilon_1 \|r\|^2.$$

(A6) We assume that the stepsize sequences $\{\gamma_k\}, \{\beta_k\}$ are positive, nonincreasing, and satisfy

$$\delta_k > 0, \; \forall k, \qquad \sum_k \delta_k = \infty, \qquad \sum_k \delta_k^2 < \infty,$$

where $\delta_k$ stands for either $\beta_k$ or $\gamma_k$. We also assume that

$$\frac{\beta_k}{\gamma_k} \to 0.$$

Note that the last assumption requires that the actor parameters be updated at a time scale slower than that of critic.

**Theorem 2.** *In an actor-critic algorithm with a TD(1) critic,*

$$\liminf_k \|\nabla\lambda(\theta_k)\| = 0 \qquad w.p. \; 1.$$

*Furthermore, if $\{\theta_k\}$ is bounded w.p. 1 then*

$$\lim_k \|\nabla\lambda(\theta_k)\| = 0 \qquad w.p. \; 1.$$

**Theorem 3.** *For every $\epsilon > 0$, there exists $\alpha$ sufficiently close to 1, such that $\liminf_k \|\nabla\lambda(\theta_k)\| \leq \epsilon$ w.p. 1.*

Note that the theoretical guarantees appear to be stronger in the case of the TD(1) critic. However, we expect that TD($\alpha$) will perform better in practice because of much smaller variance for the parameter $r_k$. (Similar issues arise when considering actor-only algorithms. The experiments reported in [7] indicate that introducing a forgetting factor $\alpha < 1$ can result in much faster convergence, with very little loss of performance.) We now provide an overview of the proofs of these theorems. Since $\beta_k/\gamma_k \to 0$, the size of the actor updates becomes negligible compared to the size of the critic updates. Therefore the actor looks stationary, as far as the critic is concerned. Thus, the analysis in [1] for the TD(1) critic and the analysis in [12] for the TD($\alpha$) critic (with $\alpha < 1$) can be used, with appropriate modifications, to conclude that the critic's approximation of $\Pi_{\theta_k} q_{\theta_k}$ will be "asymptotically correct". If $r(\theta)$ denotes the value to which the critic converges when the actor parameters are fixed at $\theta$, then the update for the actor can be rewritten as

$$\theta_{k+1} = \theta_k - \beta_k \Gamma(r(\theta_k)) Q^{\theta_k}_{r(\theta_k)}(X_{k+1}, U_{k+1})\psi_{\theta_k}(X_{k+1}, U_{k+1}) + \beta_k e_k,$$

where $e_k$ is an error that becomes asymptotically negligible. At this point, standard proof techniques for stochastic approximation algorithms can be used to complete the proof.

## 5   Conclusions

The key observation in this paper is that in actor-critic methods, the actor parameterization and the critic parameterization need not, and should not be chosen

independently. Rather, an appropriate approximation architecture for the critic is directly prescribed by the parameterization used in actor.

Capitalizing on the above observation, we have presented a class of actor-critic algorithms, aimed at combining the advantages of actor-only and critic-only methods. In contrast to existing actor-critic methods, our algorithms apply to high-dimensional problems (they do not rely on lookup table representations), and are mathematically sound in the sense that they possess certain convergence properties.

**Acknowledgments:**   This research was partially supported by the NSF under grant ECS-9873451, and by the AFOSR under grant F49620-99-1-0320.

# References

[1] D. P. Bertsekas and J. N. Tsitsiklis. *Neurodynamic Programming*. Athena Scientific, Belmont, MA, 1996.

[2] V. S. Borkar. Stochastic approximation with two time scales. *Systems and Control Letters*, 29:291–294, 1996.

[3] X. R. Cao and H. F. Chen. Perturbation realization, potentials, and sensitivity analysis of Markov processes. *IEEE Transactions on Automatic Control*, 42:1382–1393, 1997.

[4] P. W. Glynn. Stochastic approximation for monte carlo optimization. In *Proceedings of the 1986 Winter Simulation Conference*, pages 285–289, 1986.

[5] T. Jaakola, S. P. Singh, and M. I. Jordan. Reinforcement learning algorithms for partially observable Markov decision problems. In *Advances in Neural Information Processing Systems*, volume 7, pages 345–352, San Francisco, CA, 1995. Morgan Kaufman.

[6] V. R. Konda and V. S. Borkar. Actor-critic like learning algorithms for Markov decision processes. *SIAM Journal on Control and Optimization*, 38(1):94–123, 1999.

[7] P. Marbach. *Simulation based optimization of Markov reward processes*. PhD thesis, Massachusetts Institute of Technology, 1998.

[8] P. Marbach and J. N. Tsitsiklis. Simulation-based optimization of Markov reward processes. Submitted to IEEE Transactions on Automatic Control.

[9] R. Sutton and A. Barto. *Reinforcement Learning: An Introduction*. MIT Press, Cambridge, MA, 1995.

[10] R. S. Sutton, D. McAllester, S. Singh, and Y. Mansour. Policy gradient methods for reinforcement learning with function approximation. In *this proceedings*.

[11] J. N. Tsitsiklis and B. Van Roy. An analysis of temporal-difference learning with function approximation. *IEEE Transactions on Automatic Control*, 42(5):674–690, 1997.

[12] J. N. Tsitsiklis and B. Van Roy. Average cost temporal-difference learning. *Automatica*, 35(11):1799–1808, 1999.

[13] R. Williams. Simple statistical gradient following algorithms for connectionist reinforcement learning. *Machine Learning*, 8:229–256, 1992.